# Multi-Task Learning for Stock Selection

**Joumana Ghosn**
Dept. Informatique et
Recherche Opérationnelle
Université de Montréal
Montreal, Qc H3C-3J7
ghosn@iro.umontreal.ca

**Yoshua Bengio** *
Dept. Informatique et
Recherche Opérationnelle
Université de Montréal
Montreal, Qc H3C-3J7
bengioy@iro.umontreal.ca

## Abstract

Artificial Neural Networks can be used to predict future returns of stocks in order to take financial decisions. Should one build a separate network for each stock or share the same network for all the stocks? In this paper we also explore other alternatives, in which some layers are shared and others are not shared. When the prediction of future returns for different stocks are viewed as different tasks, sharing some parameters across stocks is a form of multi-task learning. In a series of experiments with Canadian stocks, we obtain yearly returns that are **more than 14% above various benchmarks**.

## 1 Introduction

Previous applications of ANNs to financial time-series suggest that several of these prediction and decision-taking tasks present sufficient non-linearities to justify the use of ANNs (Refenes, 1994; Moody, Levin and Rehfuss, 1993). These models can incorporate various types of explanatory variables: so-called technical variables (depending on the past price sequence), micro-economic stock-specific variables (such as measures of company profitability), and macro-economic variables (which give information about the business cycle).

One question addressed in this paper is whether the way to treat these different variables should be different for different stocks, i.e., should one use the same network for all the stocks or a different network for each stock? To explore this question

we performed a series of experiments in which different subsets of parameters are shared across the different stock models. When the prediction of future returns for different stocks are viewed as **different tasks** (which may nonetheless have something in common), sharing some parameters across stocks is a form of **multi-task learning**.

These experiments were performed on 9 years of data concerning 35 large capitalization companies of the Toronto Stock Exchange (TSE). Following the results of previous experiments (Bengio, 1996), the networks were not trained to predict the future return of stocks, but instead to directly optimize a financial criterion. This has been found to yield returns that are significantly superior to training the ANNs to minimize the mean squared prediction error.

In section 2, we review previous work on multi-task. In section 3, we describe the financial task that we have considered, and the experimental setup. In section 4, we present the results of these experiments. In section 5, we propose an extension of this work in which the models are re-parameterized so as to automatically learn what must be shared and what need not be shared.

## 2   Parameter Sharing and Multi-Task Learning

Most research on ANNs has been concerned with *tabula rasa* learning. The learner is given a set of examples $(x_1, y_1), (x_2, y_2), ..., (x_N, y_N)$ chosen according to some unknown probability distribution. Each pair $(x, y)$ represents an input $x$, and a desired value $y$. One defines a training criterion $C$ to be minimized in function of the desired outputs and of the outputs of the learner $f(x)$. The function $f$ is parameterized by the parameters of the network and belongs to a set of hypotheses $H$, that is the set of all functions that can be realized for different values of the parameters. The part of generalization error due to variance (due to the specific choice of training examples) can be controlled by making strong assumptions on the model, i.e., by choosing a small hypotheses space $H$ . But using an incorrect model also worsens performance.

Over the last few years, methods for automatically choosing $H$ based on similar tasks have been studied. They consider that a learner is embedded in a world where it faces **many related tasks** and that the knowledge acquired when learning a task can be used to learn better and/or faster a new task. Some methods consider that the related tasks are not always all available at the same time (Pratt, 1993; Silver and Mercer, 1995): knowledge acquired when learning a previous task is transferred to a new task. Instead, all tasks may be learned in parallel (Baxter, 1995; Caruana, 1995), and this is the approach followed here. Our objective is not to use multi-task learning to improve the speed of learning the training data (Pratt, 1993; Silver and Mercer, 1995), but instead to improve generalization performance. For example, in  (Baxter, 1995), several neural networks (one for each task) are trained simultaneouly. The networks share their first hidden layers, while all the remaining layers are specific to each network. The shared layers use the knowledge provided from the training examples of all the tasks to build an internal representation suitable for all these tasks. The remaining layers of each network use the internal representation to learn a specific task.

In the multitask learning method used by Caruana (Caruana, 1995), all the hidden

layers are shared. They serve as mutual sources of inductive bias. It was also suggested that besides the relevant tasks that are used for learning, it may be possible to use other related tasks that we do not want to learn but that may help to further bias the learner (Caruana, Baluja and Mitchell, 1996; Intrator and Edelman, 1996).

In the family discovery method (Omohundro, 1996), a parameterized family of models is built. Several learners are trained separately on different but related tasks and their parameters are used to construct a manifold of parameters. When a new task has to be learned, the parameters are chosen so as to maximize the data likelihood on the one hand, and to maximize a "family prior" on the other hand which restricts the chosen parameters to lie on the manifold.

In all these methods, the values of some or all the parameters are constrained. Such models restrict the size of the hypotheses space sufficiently to ensure good generalization performance from a small number of examples.

## 3   Application to Stock Selection

We apply the ideas of multi-task learning to a problem of stock selection and portfolio management. We consider a universe of 36 assets, including 35 risky assets and one risk-free asset. The risky assets are 35 Canadian large-capitalization stocks from the Toronto Stock Exchange. The risk-free asset is represented by 90-days Canadian treasury bills. The data is monthly and spans 8 years, from February 1986 to January 1994 (96 months). Each month, one can buy or sell some of these assets in such a way as to distribute the current worth between these assets. We do not allow borrowing or short selling, so the weights of the resulting portfolio are all non-negative (and they sum to 1).

We have selected 5 explanatory variables, 2 of which represent macro-economic variables which are known to influence the business cycle, and 3 of which are micro-economic variables representing the profitability of the company and previous price changes of the stock. The macro-economic variables were derived from yields of long-term bonds and from the Consumer Price Index. The micro-economic variables were derived from the series of dividend yields and from the series of ratios of stock price to book value of the company. Spline **extrapolation** (not interpolation) was used to obtain monthly data from the quarterly or annual company statements or macro-economic variables. For these variables, we used the dates at which their value was made public, not the dates to which they theoretically refer.

To take into account the non-stationarity of the financial and economic time-series, and estimate performance over a variety of economic situations, multiple training experiments were performed on different training windows, each time testing on the following 12 months. For each architecture, 5 such trainings took place, with training sets of size 3, 4, 5, 6, and 7 years respectively. Furthermore, multiple such experiments with different initial weights were performed to verify that we did not obtain "lucky" results due to particular initial weights. The 5 concatenated test periods make an overall 5-year test period from February 1989 to January 1994.

The training algorithm is described in (Bengio, 1996) and is based on the optimization of the neural network parameters with respect to a financial criterion (here maximizing the overall profit). The outputs of the neural network feed a trading

module. The trading module has as input at each time step the output of the network, as well as, the weights giving the current distribution of worth between the assets. These weights depend on the previous portfolio weights and on the relative change in value of each asset (due to different price changes). The outputs of the trading module are the current portfolio weights for each of the assets. Based on the difference between these desired weights and the current distribution of worth, transactions are performed. Transaction costs of 1% (of the absolute value of each buy or sell transaction) are taken into account. Because of transaction costs, the actions of the trading module at time $t$ influence the profitability of its future actions. The financial criterion depends in a non-additive way on the performance of the network over the whole sequence. To obtain gradients of this criterion with respect to the network output we have to backpropagate gradients backward through time, through the trading module, which computes a differentiable function of its inputs. Therefore, a gradient step is performed only after presenting the whole training sequence (in order, of course). In (Bengio, 1996), we have found this procedure to yield significantly larger profits (around 4% better annual return), at comparable risks, in comparison to training the neural network to predict expected future returns with the mean squared error criterion. In the experiments, the ANN was trained for 120 epochs.

## 4 Experimental Results

Four sets of experiments with different types of parameter sharing were performed, with two different architectures for the neural network: 5-3-1 (5 inputs, a hidden layer of 3 units, and 1 output), 5-3-2-1 (5 inputs, 3 units in the first hidden layer, 2 units in the second hidden layer, and 1 output). The output represents the belief that the value of the stock is going to increase (or the expected future return over three months when training with the MSE criterion).

Four types of parameter sharing between the different models for each stock are compared in these experiments: sharing everything (the same parameters for all the stocks), sharing only the parameters (weights and biases) of the first hidden layers, sharing only the output layer parameters, and not sharing anything (independent models for each stock).

The main results for the test period, using the 5-3-1 architecture, are summarized in Table 1, and graphically depicted in Figure 1 with the worth curves for the four types of sharing. The results for the test period, using the 5-3-2-1 architecture are summarized in Table 2. The ANNs were compared to two benchmarks: a buy-and-hold benchmark (with uniform initial weights over all 35 stocks), and the TSE300 Index. Since the buy-and-hold benchmark performed better (8.3% yearly return) than the TSE300 Index (4.4% yearly return) during the 02/89-01/94 test period, Tables 1, and 2 give comparisons with the buy-and-hold benchmark. Variations of average yearly return on the test period due to different initial weights were computed by performing each of the experiments 18 times with different random seeds. The resulting standard deviations are less than 3.7 when no parameters or all the parameters are shared, less than 2.7 when the parameters of the first hidden layers are shared, and less than 4.2 when the output layer is shared.

The values of **beta** and **alpha** are computed by fitting the monthly return of the portfolio $r_p$ to the return of the benchmark $r_M$, both adjusted for the risk-free return

Table 1: Comparative results for the 5-3-1 architecture: four types of sharing are compared with the buy-and-hold benchmark (see text).

|  | buy & hold | share all | share hidden | share output | no sharing |
|---|---|---|---|---|---|
| Average yearly return | 8.3% | 13% | 23.4% | 24.8% | 22.8% |
| Standard deviation (monthly) | 3.5% | 4.3% | 5.3% | 5.3% | 5.2% |
| Beta | 1 | 1.07 | 1.30 | 1.26 | 1.26 |
| Alpha (yearly) | 0 | 9% | 20.6% | 21.8% | 19.9% |
| t-statistic for alpha = 0 | NA | 11 | 14.9 | 15 | 14 |
| Reward to variability | 0.9% | 9.6% | 22.9% | 24.7% | 22.3% |
| Excess return above benchmark | 0 | 4.7% | 15.1% | 16.4% | 14.5% |
| Maximum drawdown | 15.7% | 13.3% | 13.4% | 13.3% | 13.3% |

Table 2: Comparative results for the 5-3-2-1 architecture: three types of sharing are compared with the buy-and-hold benchmark (see text).

|  | buy & hold | share all | share first hidden | share all hidden | no sharing |
|---|---|---|---|---|---|
| Average yearly return | 8.3% | 12.5% | 22.7% | 23% | 9.1% |
| Standard deviation (monthly) | 3.5% | 4.% | 5.2% | 5.2% | 3.1% |
| Beta | 1 | 1.02 | 1.25 | 1.28 | 0.87 |
| Alpha (yearly) | 0 | 8.2% | 19.7% | 20.1% | 4.% |
| t-statistic for alpha = 0 | NA | 12.1 | 14.1 | 14.8 | 21.2 |
| Reward to variability | 0.9% | 9.3% | 22.2% | 22.5% | 2.5% |
| Excess return above benchmark | 0 | 4.2% | 14.4% | 14.7% | 0.8% |
| Maximum drawdown | 15.7% | 13% | 12.6% | 13.4% | 10% |

$r_i$ (interest rates), according to the linear regression $E(r_p - r_i) = \text{alpha} + \text{beta}(r_M - r_i)$. Beta is simply the ratio of the covariance between the portfolio return and the market return with the variance of the market. According to the Capital Asset Pricing Model (Sharpe, 1964), beta gives a measure of "systematic" risk, i.e., as it relates to the risk of the market, whereas the variance of the return gives a measure of total risk. The value of alpha in the tables is annualized (as a compound return): it represents a measure of excess return (over the market benchmark) adjusted for market risk (beta). The hypothesis that alpha = 0 is clearly rejected in all cases (with t-statistics above 9, and corresponding p-values very close to 0). The **reward to variability** (or "Sharpe ratio") as defined in (Sharpe, 1966), is another risk-adjusted measure of performance: $\frac{(r_p - r_i)}{\sigma_p}$, where $\sigma_p$ is the standard deviation of the portfolio return (monthly returns were used here). The excess return above benchmark is the simple difference (not risk-adjusted) between the return of the portfolio and that of the benchmark. The maximum drawdown is another measure of risk, and it can be defined in terms of the worth curve: worth[$t$] is the ratio between the value of the portfolio at time $t$ and its value at time 0. The maximum drawdown is then defined as $\max_t \frac{((\max_{s \leq t} \text{worth}[s]) - \text{worth}[t])}{(\max_{s \leq t} \text{worth}[s])}$.

Three conclusions clearly come out of the tables and figure: (1) The main improvement is obtained by allowing some parameters to be not shared (for the 5-3-1 architecture, although the best results are obtained with a shared hidden and a free output layer, there are no significant differences between the different types of partial sharing, or no sharing at all). (2) Sharing some parameters yielded more consistent results (across architectures) than when not sharing at all. (3) The performance obtained in this way is very much better than that obtained by the benchmarks (buy-and-hold or TSE300), i.e., the yearly return is more than 14% above the best benchmark, while the risks are comparable (as measured by standard deviation of

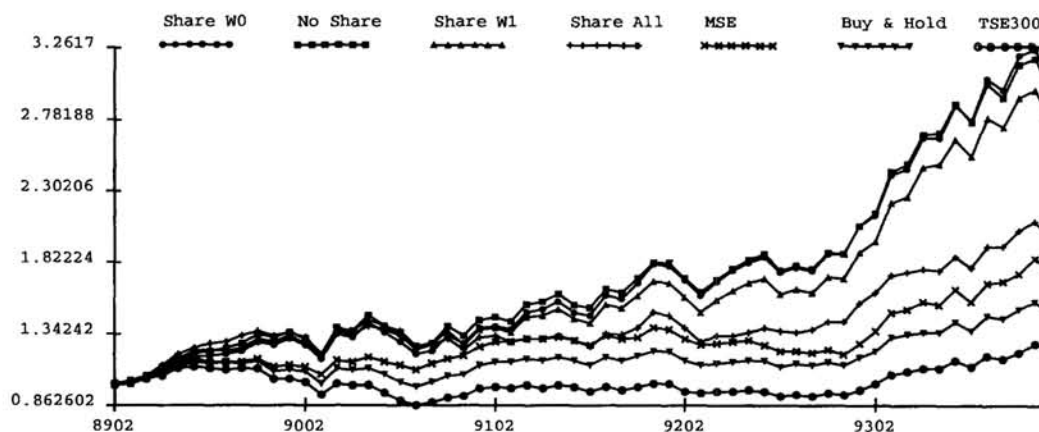

Figure 1: Evolution of total worth in the 5-year test period 02/89-01/94, for the 5-3-1 architecture, and different types of sharing. From top to bottom: sharing the hidden layer, no sharing across stocks, sharing the output layer, sharing everything, sharing everything with MSE training, Buy and Hold benchmark, TSE300 benchmark.

return or by maximum drawdown).

## 5  Future Work

We will extend the results presented here in two directions. Firstly, given the impressive results obtained with the described approach, we would like to repeat the experiment on different data sets, for different markets. Secondly, we would like to generalize the type of multi-task learning by allowing for more freedom in the way the different tasks influence each other.

Following (Omohundro, 1996), the basic idea is to re-parameterize the parameters $\theta_i \in R^{n_1}$ of the $i^{\text{th}}$ model, for all $n$ models in the following way: $\theta_i = f(p_i, \omega)$ where $p_i \in R^{n_2}$, $\omega \in R^{n_3}$, and $n \times n_1 < n \times n_2 + n_3$. For example, if $f()$ is an affine function, this forces the parameters of each the $n$ different networks to lie on the same linear manifold. The position of a point on the manifold is given by a $n_2$-dimensional vector $p_i$, and the manifold itself is specified by the $n_3$ parameters of $\omega$. The expected advantage of this approach with respect to the one used in this paper is that different models (e.g., corresponding to different stocks) may "share" more or less depending on how far their $p_i$ is from the $p_j$'s for other models. One does not have to specify which parameters are free and which are shared, but one has to specify how many are really free ($n_2$) per model, and the shape of the manifold.

## 6  Conclusion

The results presented of this paper show an interesting application of the ideas of multi-task learning to stock selection. In this paper we have addressed the question of whether ANNs trained for stock selection or portfolio management should be different for each stock or shared across all the stocks. We have found significantly better results when some or (sometimes) all of the parameters of the stock models are free (not shared). Since a parcimonuous model is always preferable, we conclude that partially sharing the parameters is even preferable, since it does not

yield a deterioration in performance, and it yields more consistent results. Another interesting conclusion of this paper is that very large returns can be obtained at risks comparable to the market using a combination of partial parameter sharing and training with respect to a financial training criterion, with a small number of explanatory input features that include technical, micro-economic and macro-economic information.

## Footnotes

*also, AT&T Labs, Holmdel, NJ 07733

# References

Baxter, J. (1995). Learning internal representations. In *Proceedings of the Eighth International Conference on Computational Learning Theory*, pages 311–320, Santa Cruz, California. ACM Press.

Bengio, Y. (1996). Using a financial training criterion rather than a prediction criterion. Technical Report #1019, Dept. Informatique et Recherche Operationnelle, Universite de Montreal.

Caruana, R. (1995). Learning many related tasks at the same time with backpropagation. In Tesauro, G., Touretzky, D. S., and Leen, T. K., editors, *Advances in Neural Information Processing Systems*, volume 7, pages 657–664, Cambridge, MA. MIT Press.

Caruana, R., Baluja, S., and Mitchell, T. (1996). Using the future to "sort out" the present: Rankprop and multitask learning for medical risk evaluation. In *Advances in Neural Information Processing Systems*, volume 8.

Intrator, N. and Edelman, S. (1996). How to make a low-dimensional representation suitable for diverse tasks. *Connection Science, Special issue on Transfer in Neural Networks*. to appear.

Moody, J., Levin, U., and Rehfuss, S. (1993). Predicting the U.S. index of industrial production. *Neural Network World*, 3(6):791–794.

Omohundro, S. (1996). Family discovery. In Mozer, M., Touretzky, D., and Perrone, M., editors, *Advances in Neural Information Processing Systems 8*. MIT Press, Cambridge, MA.

Pratt, L. Y. (1993). Discriminability-based transfer between neural networks. In Giles, C. L., Hanson, S. J., and Cowan, J., editors, *Advances in Neural Information Processing Systems 5*, pages 204–211, San Mateo, CA. Morgan Kaufmann.

Refenes, A. (1994). Stock performance modeling using neural networks: a comparative study with regression models. *Neural Networks*, 7(2):375–388.

Sharpe, W. (1964). Capital asset prices: A theory of market equilibrium under conditions of risk. *Journal of Finance*, 19:425–442.

Sharpe, W. (1966). Mutual fund performance. *Journal of Business*, 39(1):119–138.

Silver, D. L. and Mercer, R. E. (1995). Toward a model of consolidation: The retention and transfer of neural net task knowledge. In *Proceedings of the INNS World Congress on Neural Networks*, volume 3, pages 164–169, Washington, DC.
